# Rational Parametrizations of Neural Networks

**Uwe Helmke**
Department of Mathematics
University of Regensburg
Regensburg 8400 Germany

**Robert C. Williamson**
Department of Systems Engineering
Australian National University
Canberra 2601 Australia

## Abstract

A connection is drawn between rational functions, the realization theory of dynamical systems, and feedforward neural networks. This allows us to parametrize single hidden layer scalar neural networks with (almost) arbitrary analytic activation functions in terms of strictly proper rational functions. Hence, we can solve the uniqueness of parametrization problem for such networks.

## 1  INTRODUCTION

Nonlinearly parametrized representations of functions $\phi\colon \mathbb{R} \to \mathbb{R}$ of the form

$$(1.1) \qquad \phi(x) = \sum_{i=1}^{n} c_i \sigma(x - a_i) \quad x \in \mathbb{R},$$

have attracted considerable attention recently in the neural network literature. Here $\sigma\colon \mathbb{R} \to \mathbb{R}$ is typically a sigmoidal function such as

$$(1.2) \qquad \sigma(x) = (1 + e^{-x})^{-1},$$

but other choices than (1.2) are possible and of interest. Sometimes more complex representations such as

$$(1.3) \qquad \phi(x) = \sum_{i=1}^{n} c_i \sigma(b_i x - a_i)$$

or even compositions of these are considered.

The purpose of this paper is to explore some parametrization issues regarding (1.1) and in particular to show the close connection these representations have with the standard system-theoretic realization theory for rational functions. We show how to define a generalization of (1.1) parametrized by $(A, b, c)$, where $A$ is a matrix over a field, and $b$ and $c$ are vectors. (This is made more precise below). The parametrization involves the $(A, b, c)$ being used to define a rational function. The generalized $\sigma$-representation is then defined in terms of the rational function. This connection allows us to use results available for rational functions in the study of neural-network representations such as (1.1). It will also lead to an understanding of the geometry of the space of functions.

One of the main contributions of the paper is to show how in general neural network representations are related to rational functions. In this summary all proofs have been omitted. A complete version of the paper is available from the second author.

## 2    REALIZATIONS RELATIVE TO A FUNCTION

In this section we explore the relationship between sigmoidal representations of real analytic functions $\phi \colon \mathbb{I} \to \mathbb{R}$ defined on an interval $\mathbb{I} \subset \mathbb{R}$, real rational functions defined on the complex plane $\mathbb{C}$, and the well established realization theory for linear dynamical systems

$$
\begin{aligned}
\dot{x}(t) &= Ax(t) + bu(t) \\
y(t) &= cx(t) + du(t).
\end{aligned}
$$

For standard textbooks on systems theory and realization theory we refer to [5,7].

Let $\mathbb{K}$ denote either the field $\mathbb{R}$ of real numbers or the field $\mathbb{C}$ of complex numbers. Let $\Delta \subset \mathbb{C}$ be an open and simply connected subset of the complex plane and let $\sigma \colon \Delta \to \mathbb{C}$ be an analytic function defined on $\Delta$. For example, $\sigma$ may be obtained by an analytic continuation of some sigmoidal function $\sigma \colon \mathbb{R} \to \mathbb{R}$ into the domain of holomorphy of the complex plane.

Let $T \colon \mathbb{V} \to \mathbb{V}$ be a linear operator on a finite-dimensional $\mathbb{K}$-vector space $\mathbb{V}$ such that $T$ has all its eigenvalues in $\Delta$. Let $\Gamma \subset \Delta$ be a simple closed curve, oriented in the counter-clockwise direction, enclosing all the eigenvalues of $T$ in its interior. More generally, $\Gamma$ may consist of a finite number of simple closed curves $\Gamma_k$ with interiors $\Delta'_k$ such that the union of the domains $\Delta'_k$ contains all the eigenvalues of $T$. Then the matrix valued function $\sigma(T)$ is defined as the contour integral [8, p.44]

(2.1)
$$
\sigma(T) := \frac{1}{2\pi i} \int_\Gamma \sigma(z) \, (zI - T)^{-1} \, \mathrm{d}z.
$$

Note that for each linear operator $T \colon \mathbb{V} \to \mathbb{V}$, $\sigma(T) \colon \mathbb{V} \to \mathbb{V}$ is again a linear operator on $\mathbb{V}$.

If we now make the substitution $T := xI + A$ for $x \in \mathbb{C}$ and $A \colon \mathbb{V} \to \mathbb{V}$ $\mathbb{K}$-linear, then

$$
\sigma(xI + A) = \frac{1}{2\pi i} \int_\Gamma \sigma(z) \, ((z - x)I - A)^{-1} \, \mathrm{d}z
$$

becomes a function of the complex variable $x$, at least as long as $\Gamma$ contains all the eigenvalues of $xI + A$. Using the change of variables $\xi := z - x$ we obtain

$$(2.2) \qquad \sigma(xI + A) = \frac{1}{2\pi i} \int_{\Gamma'} \sigma(x + \xi)\,(\xi I - A)^{-1}\,\mathrm{d}\xi$$

where $\Gamma' = \Gamma - x \subset \Delta$ encircles all the eigenvalues of $A$.

Given an arbitrary vector $b \in \mathbb{V}$ and a linear functional $c \colon \mathbb{V} \to \mathbb{K}$ we achieve the representation

$$(2.3) \qquad \boxed{c\sigma(xI + A)b = \frac{1}{2\pi i} \int_{\Gamma} \sigma(x + \xi)\,c(\xi I - A)^{-1}b\,\mathrm{d}\xi.}$$

Note that in (2.3) the simple closed curve $\Gamma \subset \mathbb{C}$ is arbitrary, as long as it satisfies the two conditions

$(2.4) \qquad \qquad \Gamma$ encircles all the eigenvalues of $A$

$(2.5) \qquad \qquad x + \Gamma = \{x + \xi \mid \xi \in \Gamma\} \subset \Delta.$

Let $\phi \colon \mathbb{I} \to \mathbb{R}$ be a real analytic function in a single variable $x \in \mathbb{I}$, defined on an interval $\mathbb{I} \subset \mathbb{R}$.

**Definition 2.1** *A quadruple $(A, b, c, d)$ is called a finite-dimensional $\sigma$-realization of $\phi \colon \mathbb{I} \to \mathbb{R}$ over a field of constants $\mathbb{K}$ if for all $x \in \mathbb{I}$*

$$(2.6) \qquad \qquad \phi(x) = c\sigma(xI + A)b + d$$

*holds, where the right hand side is given by (2.3) and $\Gamma$ is assumed to satisfy the conditions (2.4)–(2.5). Here $d \in \mathbb{K}$, $b \in \mathbb{V}$, and $A \colon \mathbb{V} \to \mathbb{V}$, $c \colon \mathbb{V} \to \mathbb{K}$ are $\mathbb{K}$-linear maps and $\mathbb{V}$ is a finite dimensional $\mathbb{K}$-vector space.*

**Definition 2.2** *The dimension (or degree) of a $\sigma$-realization is $\dim_{\mathbb{K}} \mathbb{V}$. The $\sigma$-degree of $\phi$, denoted $\delta_{\sigma}(\phi)$, is the minimal dimension of all $\sigma$-realizations of $\phi$. A minimal $\sigma$-realization is a $\sigma$-realization of minimal dimension $\delta_{\sigma}(\phi)$.*

$\sigma$-realizations are a straightforward extension of the system-theoretic notion of a realization of a transfer function. In this paper we will address the following specific questions concerning $\sigma$-realizations.

**Q1** What are the existence and uniqueness properties of $\sigma$-realizations?

**Q2** How can one characterize minimal $\sigma$-realizations?

**Q3** How can one compute $\delta_{\sigma}(\phi)$?

## 3    EXISTENCE OF $\sigma$-REALIZATIONS

We now consider the question of existence of $\sigma$-realizations. To set the stage, we consider the systems theory case $\sigma(x) = x^{-1}$ first. Assume we are given a formal power series

$$(3.1) \qquad \qquad \phi(x) = \sum_{i=0}^{N} \frac{\phi_i}{i!} x^i, \quad N \leq \infty,$$

and that $(A, b, c)$ is a $\sigma$-realization in the sense of definition 2.1. The Taylor expansion of $c(xI + A)^{-1}b$ at 0 is (for $A$ nonsingular)

$$(3.2) \qquad c(xI + A)^{-1}b = \sum_{i=0}^{\infty}(-1)^i cA^{-(i+1)}bx^i.$$

Thus

$$(3.3) \qquad \frac{\phi_i}{i!} = (-1)^i cA^{-(i+1)}b, \qquad i = 0, \ldots, N.$$

if and only if the expansions of (3.1) and (3.2) coincide up to order $N$. Observe [7] that

$$\Longleftrightarrow \quad \begin{aligned} &\phi(x) = c(xI + A)^{-1}b \text{ and } \dim \mathbb{V} < \infty \\ &\phi(x) \text{ is rational with } \phi(\infty) = 0. \end{aligned}$$

The possibility of solving (3.3) is now easily seen as follows. Let $\mathbb{V} = \mathbb{R}^{N+1} = \text{Map}(\{0, \ldots, N\}, \mathbb{R})$ be the finite or infinite $(N+1)$-fold product space of $\mathbb{R}$. (Here $\text{Map}(X, Y)$ denotes the set of all maps from $X$ to $Y$.) If $N$ is finite let

$$(3.4) \quad A^{-1} = -\begin{bmatrix} 0 & \cdots & 0 & 1 \\ 1 & \cdots & 0 & 0 \\ & \ddots & \vdots & \vdots \\ 0 & \cdots & 1 & 0 \end{bmatrix} \in \mathbb{R}^{(N+1)\times(N+1)},$$

$$b = (1\ 0\ \cdots\ 0)^T \in \mathbb{V}, \quad c = \left(\frac{\phi_N}{N!}, \phi_0, \phi_1, \frac{\phi_2}{2!}, \ldots, \frac{\phi_{N-1}}{(N-1)!}\right).$$

For $N = \infty$ we take $A^{-1}: \mathbb{R}^{\mathbb{N}} \to \mathbb{R}^{\mathbb{N}}$ as a shift operator

$$(3.5) \qquad \begin{aligned} & A^{-1}: \mathbb{R}^{\mathbb{N}} \to \mathbb{R}^{\mathbb{N}} \\ & A^{-1}: (x_0, x_1, \ldots) \mapsto -(0, x_0, x_1, \ldots) \end{aligned}$$

$$\text{and} \qquad b = (1, 0, \ldots), \quad c = (0, \phi_0, \phi_1, \phi_2/2!, \ldots).$$

We then have

**Lemma 3.1** *Let $\sigma(x) = \sum_i \frac{\sigma_i}{i!}x^i$ be analytic at $x = 0$ and let $(A, b, c)$ be a $\sigma$-realization of the formal power series $\phi(x) = \sum_{i=0}^N \frac{\phi_i}{i!}x^i$, $N \leq \infty$ (i.e. matching of the first $N + 1$ derivatives of $\phi(x)$ and $c\sigma(xI + A)b$ at $x = 0$). Then*

$$(3.6) \qquad \phi_i = c\sigma^{(i)}(A)b \quad for\ i = 0, \ldots, N.$$

Observe that for $\sigma(x) = x^{-1}$ we have $\sigma^{(i)}(-A) = i!(A^{-1})^{i+1}$ as before. The existence part of the realization question Q1 can now be restated as

**Q4** Given $\sigma(x) := \sum_{i=0}^{\infty} \frac{\sigma_i}{i!}x^i$ and a sequence of real numbers $(\phi_0, \ldots, \phi_N)$, does there exist an $(A, b, c)$ with

$$(3.7) \qquad \phi_i = c\sigma^{(i)}(A)b, \quad i = 0, \ldots, N?$$

Thus question Q1 is essentially a Loewner interpolation question [1, 3].

Let $\gamma_\ell = cA^\ell b$, $\ell \in \mathbb{N}_0$, and let

$$(3.8) \qquad F = \begin{bmatrix} \sigma_0 & \sigma_1 & \sigma_2 & \cdots \\ \sigma_1 & \sigma_2 & \sigma_3 & \cdots \\ \sigma_2 & \sigma_3 & \sigma_4 & \cdots \\ \vdots & \vdots & \vdots & \ddots \end{bmatrix} = (\sigma_{i+j})_{i,j=0}^{\infty}.$$

Write

$$(3.9) \qquad [\gamma] = \begin{bmatrix} \gamma_0 \\ \gamma_1 \\ \gamma_2/2! \\ \gamma_3/3! \\ \vdots \end{bmatrix}, \quad \text{and} \quad [\phi] = \begin{bmatrix} \phi_0 \\ \phi_1 \\ \phi_2 \\ \vdots \end{bmatrix}.$$

Then (3.6) (for $N = \infty$) can formally be written as

$$(3.10) \qquad [\phi] = F \cdot [\gamma].$$

Of course, any meaningful interpretation of (3.10) requires that the infinite sums $\sum_{j=0}^{\infty} \frac{\sigma_{i+j}}{j!} \gamma_j$, $i \in \mathbb{N}_0$, exist. This happens, for example, if $\sum_{j=0}^{\infty} \sigma_{i+j}^2 < \infty$, $i \in \mathbb{N}_0$ and $\sum_{j=0}^{\infty} (\gamma_j/j!)^2 < \infty$ exist. We have already seen that every finite or infinite sequence $[\gamma]$ has a realization $(A, b, c)$. Thus we obtain

**Corollary 3.2** *A function $\phi(x)$ admits a $\sigma$-realization if and only if $[\phi] \in \text{image}(F)$.*

**Corollary 3.3** *Let $H = (\gamma_{i+j})_{i,j=0}^{\infty}$. There exists a finite dimensional $\sigma$-realization of $\phi(x)$ if and only if $[\phi] = F[\gamma]$ with $\text{rank} H < \infty$. In this case $\delta_\sigma(\phi) = \text{rank} H$.*

# 4    UNIQUENESS OF $\sigma$-REALIZATIONS

In this section we consider the uniqueness of the representation (2.3).

**Definition 4.1 (c.f. [2])** *A system $\{g_1, \ldots, g_n\}$ of continuous functions $g_i \colon \mathbb{I} \to \mathbb{R}$, defined on an interval $\mathbb{I} \subset \mathbb{R}$, is said to satisfy a Haar\* condition of order $n$ on $\mathbb{I}$ if $g_1, \ldots, g_n$ are linearly independent, i.e. For every $c_1, \ldots, c_n \in \mathbb{R}$ with $\sum_{i=1}^{n} c_i g_i(x) = 0$ for all $x \in \mathbb{I}$, then $c_1 = \cdots = c_n = 0$.*

**Remark**  The Haar\* condition is implied by the stronger classical Haar condition that

$$\det \begin{bmatrix} g_1(x_1) & \cdots & g_1(x_n) \\ \vdots & & \vdots \\ g_n(x_1) & \cdots & g_n(x_n) \end{bmatrix} \neq 0$$

for all distinct $(x_i)_{i=1}^{n}$ in $\mathbb{I}$. Equivalently, if $\sum_{i=1}^{n} c_i g_i(x)$ has $n$ distinct roots in $\mathbb{I}$, then $c_1 = \cdots = c_n = 0$.

**Definition 4.2** *A subset $A$ of $\mathbb{C}$ is called self-conjugate if $a \in A$ implies $\bar{a} \in A$.*

Let $\sigma\colon \mathbb{R} \to \mathbb{R}$ be a continuous function and define $\sigma_{z_i}^{(j)}(x) := \sigma^{(j)}(x + z_i)$. Let

$$\kappa := (\kappa_1, \ldots, \kappa_m) \quad \text{where} \quad \sum_{j=1}^{m} \kappa_j = n, \ \kappa_j \in \mathbb{N}, \ \kappa_j \geq 1, \ j = 1, \ldots, m$$

denote a combination of $n$ of size $m$. For a given combination $\kappa = (\kappa_1, \ldots, \kappa_m)$ of $n$, let $I := \{1, \ldots, m\}$ and let $J_i := \{1, \ldots, \kappa_i\}$. Let $Z_m := \{z_1, \ldots, z_m\}$ and let

(4.1) $$\sigma(\kappa, Z_m) := \{\sigma_{z_i}^{(j-1)} \colon i \in I, j \in J_i\}.$$

**Definition 4.3** *If for all $m \leq n$, for all combinations $\kappa = (\kappa_1, \ldots, \kappa_m)$ of $n$ of size $m$, and for any self-conjugate set $Z_m$ of distinct points, $\sigma(\kappa, Z_m)$ satisfies a Haar* condition of order $n$, then $\sigma$ is said to be* Haar generating *of order $n$.*

**Theorem 4.4 (Uniqueness)** *Let $\sigma\colon \mathbb{R} \to \mathbb{R}$ be Haar generating of order at least $2n$ on $\mathbb{I}$ and let $(A, b, c)$ and $(\tilde{A}, \tilde{b}, \tilde{c})$ be minimal $\sigma$-realizations of order $n$ of functions $\phi$ and $\tilde{\phi}$ respectively. Then the following equivalence holds*

(4.2) $$\begin{aligned} c\sigma(xI + A)b &= \tilde{c}\sigma(xI + \tilde{A})\tilde{b} \quad \forall x \in \mathbb{I} \\[1mm] &\Longleftrightarrow \\[1mm] c(\xi I - A)^{-1}b &= \tilde{c}(\xi I - \tilde{A})^{-1}\tilde{b} \quad \forall \xi \in \mathbb{R}. \end{aligned}$$

*Conversely, if (4.2) holds for almost all order $n$ triples $(A, b, c)$, $(\tilde{A}, \tilde{b}, \tilde{c})$, then $\sigma\colon \mathbb{R} \to \mathbb{R}$ is Haar generating on $\mathbb{I}$ of order $\geq n$.*

The following result gives examples of activation functions $\sigma\colon \mathbb{R} \to \mathbb{R}$ which are Haar generating.

**Lemma 4.5** *Let $d \in \mathbb{N}_0$. Then 1) The function $\sigma(x) = x^{-d}$ is Haar generating of arbitrary order. 2) The monomial $\sigma(x) = x^d$ is Haar generating of order $d + 1$. 3) The function $e^{-x^2}$ is Haar generating of arbitrary order.*

**Remark** A simple example of a $\sigma$ which is not Haar generating of order $\geq 2$ is $\sigma(x) = e^x$. In fact, in this case $\sigma(x + z_j) = c_j\sigma(x + z_i)$ for $c_j = e^{z_j - z_1}$, $j = 2, \ldots, n$.

**Remark** The function $\sigma(x) = (1 + e^{-x})^{-1}$ is *not* Haar generating of any order $\geq 2$. By the periodicity of the complex exponential function, $\sigma(x + 2\pi i) = \sigma(x - 2\pi i)$, $i = \sqrt{-1}$, for all $x$. Thus the Haar* condition fails for $Z_2 = \{2\pi i, -2\pi i\}$.

In particular, the above uniqueness result fails for the standard sigmoid case. In order to cover this case we need a further definition.

**Definition 4.6** *Let $\Omega = \overline{\Omega} \subset \mathbb{C}$ be a self-conjugate subset of $\mathbb{C}$. A function $\sigma\colon \mathbb{R} \to \mathbb{R}$ is said to be* Haar generating *of order $n$ on $\Omega$, if for all $m \leq n$, for all combinations $\kappa = (\kappa_1, \ldots, \kappa_m)$ of $n$ of size $m$, and for any self-conjugate subset $Z_m \subset \Omega$ of distinct points of $\Omega$, $\sigma(\kappa, Z_m)$ satisfies a Haar* condition of order $n$.*

Of course for $\Omega = \mathbb{C}$, this definition coincides with definition 4.3.

**Theorem 4.7 (Local Uniqueness)** *Let $\sigma: \mathbb{R} \to \mathbb{R}$ be analytic and let $\Omega \subset \mathbb{C}$ be a self-conjugate subset contained in the domain of holomorphy of $\sigma$. Let $\mathbb{I}$ be a nontrivial subinterval of $\Omega \cap \mathbb{R}$. Suppose $\sigma: \mathbb{R} \to \mathbb{R}$ is Haar generating on $\Omega$ of order at least $2n$, $n \in \mathbb{N}$. Then for any two minimal $\sigma$-realizations $(A, b, c)$ and $(\tilde{A}, \tilde{b}, \tilde{c})$ of orders at most $n$ with $\operatorname{spect} A$, $\operatorname{spect} \tilde{A} \in \Omega$ the following equivalence holds:*

$$(4.3) \qquad \Longleftrightarrow \qquad \begin{aligned} c\sigma(xI + A)b &= \tilde{c}\sigma(xI + \tilde{A})\tilde{b} \quad \forall x \in \mathbb{I} \\ c(\xi I - A)^{-1}b &= \tilde{c}(\xi I - \tilde{A})^{-1}\tilde{b} \quad \forall \xi \in \mathbb{R}. \end{aligned}$$

**Lemma 4.8** *Let $\Omega := \{z \in \mathbb{C}: |\Im z| < \pi\}$. Then the standard sigmoid function $\sigma(x) = (1 + e^{-x})^{-1}$ is Haar generating on $\Omega$ of arbitrary order.*

## 5  MAIN RESULT

As a consequence of the uniqueness theorems 4.4 and 4.7 we can now state our main result on the existence of minimal $\sigma$-realizations of a function $\phi(x)$. It extends a parallel result for standard transfer function realizations, where $\sigma(x) = x^{-1}$.

**Theorem 5.1 (Realization)** *Let $\Omega \subset \mathbb{C}$ be a self-conjugate subset, contained in the domain of holomorphy of a real meromorphic function $\sigma: \mathbb{R} \to \mathbb{R}$. Suppose $\sigma$ is Haar generating on $\Omega$ of order at least $2n$ and assume $\phi(x)$ has a finite dimensional realization $(A, b, c)$ of dimension at most $n$ such that $A$ has all its eigenvalues in $\Omega$.*

1. *There exists a minimal $\sigma$-realization $(A_1, b_1, c_1)$ of $\phi(x)$ of degree $\delta_\sigma(\phi) \leq \dim(A, b, c)$. Furthermore, there exists an invertible matrix $S$ such that*

$$(5.1) \qquad SAS^{-1} = \begin{bmatrix} A_1 & A_2 \\ 0 & A_3 \end{bmatrix}, \quad Sb = \begin{bmatrix} b_1 \\ 0 \end{bmatrix}, \quad cS^{-1} = [c_1, c_2].$$

2. *If $(A_1, b_1, c_1)$ and $(A_1', b_1', c_1')$ are minimal $\sigma$-realizations of $\phi(x)$ such that the eigenvalues of $A_1$ and $A_1'$ are contained in $\Omega$, then there exists a unique invertible matrix $S$ such that*

$$(5.2) \qquad (A_1', b_1', c_1') = (SA_1 S^{-1}, Sb_1, c_1 S^{-1}).$$

3. *A $\sigma$-realization $(A, b, c)$ is minimal if and only if $(A, b, c)$ is controllable and observable; i.e. if and only if $(A, b, c)$ satisfies the generic rank conditions*

$$\operatorname{rank}(b, Ab, \dots, A^{n-1}b) = n, \qquad \operatorname{rank} \begin{bmatrix} c \\ cA \\ \vdots \\ cA^{n-1} \end{bmatrix} = n$$

*for $A \in \mathbb{K}^{n \times n}$, $b \in \mathbb{K}^n$, $c^T \in \mathbb{K}^n$.*

**Remark**  The use of the terms "observable" and "controllable" is solely for formal correspondence with standard systems theory. There are no dynamical systems actually under consideration here.

**Remark** Note that for any $\sigma$-realization $(A, b, c)$ of the form $A = \begin{bmatrix} A_{11} & A_{12} \\ 0 & A_{22} \end{bmatrix}$, $b = \begin{bmatrix} b_1 \\ 0 \end{bmatrix}$, $c = [c_1, c_2]$, we have $\sigma(A) = \begin{bmatrix} \sigma(A_{11}) & * \\ 0 & \sigma(A_{22}) \end{bmatrix}$ and thus $c\sigma(xI + A)b = c_1\sigma(xI + A_{11})b_1$. Thus transformations of the above kind always *reduce* the dimension of a $\sigma$-realization.

**Corollary 5.2 ([9])** *Let* $\sigma(x) = (1 + e^{-x})^{-1}$ *and let* $\phi(x) = \sum_{i=1}^{n} c_i\sigma(x - a_i) = \sum_{i=1}^{n} c_i'\sigma(x - a_i')$ *be two minimal length $\sigma$-representations with* $|\Im a_i| < \pi$, $|\Im a_i'| < \pi$, $i = 1, \ldots, n$. *Then* $(a_i', c_i') = (a_{p(i)}, c_{p(i)})$ *for a unique permutation* $p\colon \{1, \ldots, n\} \to \{1, \ldots, n\}$. *In particular, minimal length representation (1.1) with real coefficients $a_i$ and $c_i$ are unique up to a permutation of the summands.*

## 6    CONCLUSIONS

We have drawn a connection between the realization theory for linear dynamical systems and neural network representations. There are further connections (not discussed in this summary) between representations of the form (1.3) and rational functions of two variables. There are other questions concerning diagonalizable realizations and Jordan forms. Details are given in the full length version of this paper. Open questions include the problem of partial realizations [4,6].[1]

## REFERENCES

[1]  A. C. Antoulas and B. D. O. Anderson, On the Scalar Rational Interpolation Problem, *IMA Journal of Mathematical Control and Information*, **3** (1986), pp. 61–88.

[2]  E. W. Cheney, *Introduction to Approximation Theory*, Chelsea Publishing Company, New York, 1982.

[3]  W. F. Donoghue, Jr, *Monotone Matrix Functions and Analytic Continuation*, Springer-Verlag, Berlin, 1974.

[4]  W. B. Gragg and A. Lindquist, On the Partial Realization Problem, *Linear Algebra and its Applications*, **50** (1983), pp. 277–319.

[5]  T. Kailath, *Linear Systems*, Prentice-Hall, Englewood Cliffs, 1980.

[6]  R. E. Kalman, On Partial Realizations, Transfer Functions, and Canonical Forms, *Acta Polytechnica Scandinavica*, **31** (1979), pp. 9–32.

[7]  R. E. Kalman, P. L. Falb and M. A. Arbib, *Topics in Mathematical System Theory*, McGraw-Hill, New York, 1969.

[8]  T. Kato, *Perturbation Theory for Linear Operators*, Springer-Verlag, Berlin, 1966.

[9]  R. C. Williamson and U. Helmke, Existence and Uniqueness Results for Neural Network Approximations, To appear, IEEE Transactions on Neural Networks, 1993.

[1]This work was supported by the Australian Research Council, the Australian Telecommunications and Electronics Research Board, and the Boeing Commercial Aircraft Company (thanks to John Moore). Thanks to Eduardo Sontag for helpful comments also.